# Invariant Common Spatial Patterns: Alleviating Nonstationarities in Brain-Computer Interfacing

**Benjamin Blankertz**[1,2]      **Motoaki Kawanabe**[2]      **Ryota Tomioka**[3]

**Friederike U. Hohlefeld**[4]      **Vadim Nikulin**[5]

**Klaus-Robert Müller**[1,2]

[1]TU Berlin, Dept. of Computer Science, Machine Learning Laboratory, Berlin, Germany
[2]Fraunhofer FIRST (IDA), Berlin, Germany
[3]Dept. Mathematical Informatics, IST, The University of Tokyo, Japan
[4]Berlin School of Mind and Brain, Berlin, Germany
[5]Dept. of Neurology, Campus Benjamin Franklin, Charité University Medicine Berlin, Germany
`{blanker,krm}@cs.tu-berlin.de`

## Abstract

Brain-Computer Interfaces can suffer from a large variance of the subject conditions within and across sessions. For example vigilance fluctuations in the individual, variable task involvement, workload etc. alter the characteristics of EEG signals and thus challenge a stable BCI operation. In the present work we aim to define features based on a variant of the common spatial patterns (CSP) algorithm that are constructed *invariant* with respect to such nonstationarities. We enforce invariance properties by adding terms to the denominator of a Rayleigh coefficient representation of CSP such as disturbance covariance matrices from fluctuations in visual processing. In this manner physiological prior knowledge can be used to shape the classification engine for BCI. As a proof of concept we present a BCI classifier that is robust to changes in the level of parietal $\alpha$-activity. In other words, the EEG decoding still works when there are lapses in vigilance.

## 1   Introduction

Brain-Computer Interfaces (BCIs) translate the intent of a subject measured from brain signals directly into control commands, e.g. for a computer application or a neuroprosthesis ([1, 2, 3, 4, 5, 6]). The classical approach to brain-computer interfacing is *operant conditioning* ([2, 7]) where a fixed translation algorithm is used to generate a feedback signal from the electroencephalogram (EEG). Users are not equipped with a mental strategy they should use, rather they are instructed to watch a feedback signal and using the feedback to find out ways to voluntarily control it. Successful BCI operation is reinforced by a reward stimulus. In such BCI systems the user adaption is crucial and typically requires extensive training. Recently *machine learning techniques* were applied to the BCI field and allowed to decode the subject's brain signals, placing the learning task on the machine side, i.e. a general translation algorithm is trained to infer the specific characteristics of the user's brain signals [8, 9, 10, 11, 12, 13, 14]. This is done by a statistical analysis of a calibration measurement in which the subject performs well-defined mental acts like imagined movements. Here, in principle no adaption of the user is required, but it is to be expected that users will adapt their behaviour during feedback operation. The idea of the machine learning approach is that a flexible adaption of the system relieves a good amount of the learning load from the subject. Most BCI systems are somewhere between those extremes.

Although the proof-of-concept of machine learning based BCI systems[1] was given some years ago, several major challenges are still to be faced. One of them is to make the system *invariant* to non task-related fluctuations of the measured signals during feedback. These fluctuations may be caused by changes in the subject's brain processes, e.g. change of task involvement, fatigue etc., or by artifacts such as swallowing, blinking or yawning. The calibration measurement that is used for training in machine learning techniques is recorded during 10-30 min, i.e. a relatively short period of time and typically in a monotone atmosphere, so this data does not contain all possible kinds of variations to be expected during on-line operation.

The present contribution focusses on invariant feature extraction for BCI. In particular we aim to enhance the invariance properties of the common spatial patterns (CSP, [15]) algorithm. CSP is the solution of a generalized eigenvalue problem and has as such a strong link to the maximization of a Rayleigh coefficient, similar to Fisher's discriminant analysis. Prior work by Mika et al. [16] in the context of kernel Fisher's discriminant analysis contains the key idea that we will follow: noise and distracting signal aspects with respect to which we want to make our feature extractor invariant is added to the denominator of a Rayleigh coefficient. In other words, our prior knowledge about the noise type helps to re-design the optimization of CSP feature extraction. We demonstrate how our invariant CSP (iCSP) technique can be used to make a BCI system invariant to changes in the power of the parietal $\alpha$-rhythm (see Section 2) reflecting, e.g. changes in vigilance. Vigilance changes are among the most pressing challenges when robustifying a BCI system for long-term real-world applications.

In principle we could also use an adaptive BCI, however, adaptation typically has a limited time scale which might not allow to follow fluctuations quickly enough. Furthermore online adaptive BCI systems have so far only been operated with 4-9 channels. We would like to stress that adaptation and invariant classification are no mutually exclusive alternatives but rather complementary approaches when striving for the same goal: a BCI system that is invariant to undesired distortions and non-stationarities.

## 2 Neurophysiology and Experimental Paradigms

**Neurophysiological background.** Macroscopic brain activity during resting wakefulness contains distinct 'idle' rhythms located over various brain areas, e.g. the parietal $\alpha$-rhythm (7-13 Hz) can be measured over the visual cortex [17] and the $\mu$-rhythm can be measured over the pericentral sensorimotor cortices in the scalp EEG, usually with a frequency of about 8–14 Hz ([18]). The strength of the parietal $\alpha$-rhythm reflects visual processing load as well as attention and fatigue resp. vigilance.

The moment-to-moment amplitude fluctuations of these local rhythms reflect variable functional states of the underlying neuronal cortical networks and can be used for brain-computer interfacing. Specifically, the pericentral $\mu$- and $\beta$ rythms are diminished, or even almost completely blocked, by movements of the somatotopically corresponding body part, independent of their active, passive or reflexive origin. Blocking effects are visible bilateral but with a clear predominance contralateral to the moved limb. This attenuation of brain rhythms is termed event-related desynchronization (ERD) and the dual effect of enhanced brain rhythms is called event-related synchronization (ERS) (see [19]).

Since a focal ERD can be observed over the motor and/or sensory cortex even when a subject is only imagining a movement or sensation in the specific limb, this feature can be used for BCI control: The discrimination of the imagination of movements of left hand vs. right hand vs. foot can be based on the somatotopic arrangement of the attenuation of the $\mu$ and/or $\beta$ rhythms. However the challenge is that due to the volume conduction EEG signal recorded at the scalp is a mixture of many cortical activities that have different spatial localizations; for example, at the electrodes over the mortor cortex, the signal not only contains the $\mu$-rhythm that we are interested in but also the projection of parietal $\alpha$-rhythm that has little to do with the motor-imagination. To this end, *spatial filtering* is an indispensable technique; that is to take a linear combination of signals recorded over EEG channels and extract only the component that we are interested in. In particular the CSP algorithm that optimizes spatial filters with respect to discriminability is a good candidate for feature extraction.

**Experimental Setup.** In this paper we evaluate the proposed algorithm on off-line data in which the nonstationarity is induced by having two different background conditions for the same primary

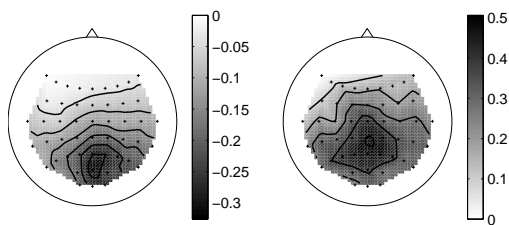

Figure 1: Topographies of $r^2$–values (multiplied by the sign of the difference) quantifying the difference in log band-power in the alpha band (8–12 Hz) between different recording sessions: *Left:* Difference between *imag_move* and *imag_lett*. Due to lower visual processing demands, alpha power in occipital areas is stronger in *imag_lett*. *Right:* Difference between *imag_move* and *sham_feedback*. The latter has decreased alpha power in centro-parietal areas. Note the different sign in the colormaps.

task. The ultimate challenge will be on-line feedback with strong fluctuations of task demands etc, a project envisioned for the near future.

We investigate EEG recordings from 4 subjects (all from whom we have an 'invariance measurement', see below). Brain activity was recorded from the scalp with multi-channel amplifiers using 55 EEG channels.

In the 'calibration measurement' all 4.5–6 seconds one of 3 different visual stimuli indicated for 3 seconds which mental task the subject should accomplish during that period. The investigated mental tasks were imagined movements of the left hand, the right hand, and the right foot. There were two types of visual stimulation: (1: *imag_lett*) targets were indicated by letters (L, R, F) appearing at a central fixation cross and (2: *imag_move*) a randomly moving small rhomboid with either its left, right or bottom corner filled to indicate left or right hand or foot movement, respectively. Since the movement of the object was independent from the indicated targets, target-uncorrelated eye movements are induced. Due to the different demands in visual processing, the background brain activity can be expected to differ substancially in those two types of recordings. The topography of the $r^2$–values (bi-serial correlation coefficient of feature values with labels) of the log band-power difference between *imag_move* and *imag_lett* is shown in the left plot of Fig. 2. It shows a pronounced differene in parietal areas.

A *sham_feedback* paradigm was designed in order to charaterize invariance properties needed for stable real-world BCI applications. In this measurement the subjects received a fake feedback sequence which was preprogrammed. The aim of this recording was to collect data during a large variety of mental states and actions that are *not* correlated with the BCI control states (motor imagery of hands and feet). Subjects were told that they could control the feedback in some way that they should find out, e.g. with eye movements or muscle activity. They were instructed not to perform movements of hands, arms, legs and feet. The type of feedback was a standard 1D cursor control. In each trial the cursor starts in the middle and should be moved to either the left or right side as indicated by a target cue. When the cursor touched the left or right border, a response (correct or false) was shown. Furthermore the number of hits and misses was shown. The preprogrammed 'feedback' signal was constructed such that it was random in the beginning and then alternating periods of increasingly more hits and periods with chance level performance. This was done to motivate the subjects to try a variety of different actions and to induce different states of mood (satisfaction during 'successful' periods and anger resp. disfavor during 'failure'). The right plot of Fig. 2 visualizes the difference in log band-power between *imag_move* and *sham_feedback*. A decreased alpha power in centro-parietal areas during *sham_feedback* can be observed. Note that this recording includes much more variations of background mental activity than the difference between *imag_move* and *imag_lett*.

## 3 Methods

**Common Spatial Patterns (CSP) Analysis.**  The CSP technique ([15]) allows to determine spatial filters that maximize the variance of signals of one condition and at the same time minimize the variance of signals of another condition. Since variance of band-pass filtered signals is equal to band-power, CSP filters are well suited to discriminate mental states that are characterized by ERD/ERS effects ([20]). As such it has been well used in BCI systems ([8, 14]) where CSP filters are calculated individually for each subject on the data of a calibration measurement.

Technically the Common Spatial Pattern (CSP) [21] algorithm gives spatial filters based on a discriminative criterion. Let $X_1$ and $X_2$ be the (time $\times$ channel) data matrices of the band-pass filtered

EEG signals (concatenated trials) under the two conditions (e.g., right-hand or left-hand imagination, respectively[2]) and $\Sigma_1$ and $\Sigma_2$ be the corresponding estimates of the covariance matrices $\Sigma_i = X_i^\top X_i$. We define the two matrices $S_d$ and $S_c$ as follows:

$$S_d = \Sigma^{(1)} - \Sigma^{(2)} \qquad : \text{discriminative activity matrix,}$$

$$S_c = \Sigma^{(1)} + \Sigma^{(2)} \qquad : \text{common activity matrix.}$$

The CSP spatial filter $v \in \mathbb{R}^C$ ($C$ is the number of channels) can be obtained by extremizing the Rayleigh coefficient:

$$\{\max, \min\}_{v \in \mathbb{R}^C} \quad \frac{v^\top S_d v}{v^\top S_c v}. \tag{1}$$

This can be done by solving a generalized eigenvalue problem.

$$S_d v = \lambda S_c v. \tag{2}$$

The eigenvalue $\lambda$ is bounded between $-1$ and $1$; a large positive eigenvalue corresponds to a projection of the signal given by $v$ that has large power in the first condition but small in the second condition; the converse is true for a large negative eigenvalue. The largest and the smallest eigenvalues correspond to the maximum and the minimum of the Rayleigh coefficient problem (Eq. (1)). Note that $v^\top S_d v = v^\top \Sigma_1 v - v^\top \Sigma_2 v$ is the average power difference in two conditions that we want to maximize. On the other hand, the projection of the activity that is common to two classes $v^\top S_c v$ should be minimized because it doesn't contribute to the discriminability. Using the same idea from [16] we can rewrite the Rayleigh problem (Eq. (1)) as follows:

$$\min_{v \in \mathbb{R}^C} \quad v^\top S_c v, \qquad \text{s.t.} \quad v^\top \Sigma_1 v - v^\top \Sigma_2 v = \lambda,$$

which can be interpreted as finding the minimum norm $v$ with the condition that the average power difference between two conditions to be $\lambda$. The norm is defined by the common activity matrix $S_c$. In the next section, we extend the notion of $S_c$ to incorporate any disturbances that is common to two classes that we can measure a priori.

In this paper we call *filter* the generalized eigenvectors $v_j$ ($j = 1, \ldots, C$) of the generalized eigenvalue problem (Eq. (2)) or a similar problem discussed in the next section. Moreover we denote by $V$ the matrix we obtain by putting the $C$ generalized eigenvectors into columns, namely $V = \{v_j\}_{j=1}^C \in \mathbb{R}^{C \times C}$ and call *patterns* the row vectors of the inverse $A = V^{-1}$. Note that a filter $v_j \in \mathbb{R}^C$ has its corresponding pattern $a_j \in \mathbb{R}^C$; a filter $v_j$ extracts only the activity spanned by $a_j$ and cancels out all other activities spanned by $a_i$ ($i \neq j$); therefore a pattern $a_j$ tells what the filter $v_j$ is extracting out (see Fig. 2).

For classification the features of single-trials are calculated as the log-variance in CSP projected signals. Here only a few (2 to 6) patterns are used. The selection of patterns is typically based on eigenvalues. But when a large amount of calibration data is not available it is advisable to use a more refined technique to select the patterns or to manually choose them by visual inspection. The variance features are approximately chi-square distributed. Taking the logarithm makes them similar to gaussian distributions, so a linear classifier (e.g., linear discriminant analysis) is fine.

For the evaluation in this paper we used the CSPs corresponding the the two largest and the two smallest eigenvalues and used linear disciminant analysis for classification. The CSP algorithm, several extentions as well as practical issues are reviewed in detail in [15].

**Invariant CSP.** The CSP spatial filters extracted as above are optimized for the calibration measurement. However, in online operation of the BCI system different non task-related modulations of brain signals may occur which are not suppressed by the CSP filters. The reason may be that these modulations have not been recorded in the calibration measurement or that they have been so infrequent that they are not consistently reflected in the statistics (e.g. when they are not equally distributed over the two conditions).

The proposed iCSP method minimizes the influence of modulations that can be characterized in advance by a covariance matrix. In this manner we can code neurophysiological prior knowledge

or further information such as the tangent covariance matrix ([22]) into such a covariante matrix $\Xi$. In the following motivation we assume that $\Xi$ is the covariance matrix of a signal matrix $Y$. Using the notions from above, the objective is then to calculate spatial filters $v_j^{(1)}$ such that $\mathrm{var}(X_1 v_j^{(1)})$ is maximized and $\mathrm{var}(X_2 v_j^{(1)})$ and $\mathrm{var}(Y v_j^{(1)})$ are minimized. Dually spatial filters $v_j^{(2)}$ are determined that maximize $\mathrm{var}(X_2 v_j^{(2)})$ and minimize $\mathrm{var}(X_1 v_j^{(2)})$ and $\mathrm{var}(Y v_j^{(2)})$.

Pratically this can be accomplished by solving the following two generalized eigenvalue problems:

$$V^{(1)^\top} \Sigma_1 V^{(1)} = D^{(1)} \quad \text{and} \quad V^{(1)^\top}((1-\xi)(\Sigma_1 + \Sigma_2) + \xi\Xi)V^{(1)} = I \tag{3}$$

$$V^{(2)^\top} \Sigma_2 V^{(2)} = D^{(2)} \quad \text{and} \quad V^{(2)^\top}((1-\xi)(\Sigma_1 + \Sigma_2) + \xi\Xi)V^{(2)} = I \tag{4}$$

where $\xi \in [0,1]$ is a hyperparameter to trade-off the discrimination of the training classes ($X_1$, $X_2$) against invariance (as characterized by $\Xi$). Section 4 discusses the selection of parameter $\xi$. Filters $v_j^{(1)}$ with high eigenvalues $d_j^{(1)}$ provide not only high $\mathrm{var}(X_1 v_j^{(1)})$ but also small $v_j^{(1)^\top}((1-\xi)\Sigma_2 + \xi\Xi)v_j^{(1)} = 1 - (1-\xi)d_j^{(1)}$, i.e. small $\mathrm{var}(X_2 v_j^{(1)})$ *and* small $\mathrm{var}(Y v_j^{(1)})$. The dual is true for the selection of filters from $v_j^{(2)}$.

Note that for $\xi = 0.5$ there is a strong connection to the one-vs-rest strategy for 3-class CSP ([23]). Features for classification are calculated as log-variance using the two filters from each of $v_j^{(1)}$ and $v_j^{(2)}$ corresponding to the largest eigenvalues. Note that the idea of iCSP is in the spirit of the invariance constraints in (kernel) Fisher's Discriminant proposed in [16].

**A Theoretical Investigation of iCSP by Influence Analysis.** As mentioned, iCSP is aiming at robust spatial filtering against disturbances whose covariance $\Xi$ can be anticipated from prior knowledge. Influence analysis is a statistical tool with which we can assess robustness of inference procedures [24]. Basically, it evaluates the effect in inference procedures, if we add a small perturbation of $O(\varepsilon)$, where $\varepsilon \ll 1$. For example, influence functions for the component analyses such as PCA and CCA have been discussed so far [25, 26]. We applied the machinery to iCSP, in order to check whether iCSP really reduces influence caused by the disturbance at least in local sense. For this purpose, we have the following lemma (its proof is included in the Appendix).

**Lemma 1** (*Influence of generalized eigenvalue problems*) *Let $\lambda_k$ and $w_k$ be k-th eigenvalue and eigenvector of the generalized eigvenvalue problem*

$$Aw = \lambda Bw, \tag{5}$$

*respectively. Suppose that the matrices A and B are perturbed with small matrices $\varepsilon\Delta$ and $\varepsilon P$ where $\varepsilon \ll 1$. Then the eigenvalues $\widetilde{w}_k$ and eigenvectors $\widetilde{\lambda}_k$ of the purterbed problem*

$$(A + \varepsilon\Delta)\widetilde{w} = \widetilde{\lambda}(B + \varepsilon P)\widetilde{w} \tag{6}$$

*can be expanded as $\lambda_k + \varepsilon\chi_k + o(\varepsilon)$ and $w_k + \varepsilon\psi_k + o(\varepsilon)$, where*

$$\chi_k = w_k^\top(\Delta - \lambda_k P)w_k, \qquad \psi_k = -M_k(\Delta - \lambda_k P)w_k - \frac{1}{2}(w_k^\top P w_k)w_k, \tag{7}$$

$M_k := B^{-1/2}(B^{-1/2}AB^{-1/2} - \lambda_k I)^+ B^{-1/2}$ *and the suffix '+' denotes Moore-Penrose matrix inverse.*

The generalized eigenvalue problem eqns (3) and (4) can be rephrased as

$$\Sigma_1 v = d\{(1-\xi)(\Sigma_1 + \Sigma_2) + \xi\Xi\}v, \qquad \Sigma_2 u = c\{(1-\xi)(\Sigma_1 + \Sigma_2) + \xi\Xi\}u.$$

For simplicity, we consider here the simplest perturbation of the covariances as $\Sigma_1 \to \Sigma_1 + \varepsilon\Xi$ and $\Sigma_2 \to \Sigma_1 + \varepsilon\Xi$. In this case, the perturbation matrices in the lemma can be expressed as $\Delta_1 = \Xi$, $\Delta_2 = \Xi$, $P = 2(1-\xi)\Xi$. Therefore, we get the expansions of the eigenvalues and eigenvectors as $d_k + \varepsilon\chi_{1k}$, $c_k + \varepsilon\chi_{2k}$, $v_k + \varepsilon\psi_{1k}$ and $u_k + \varepsilon\psi_{2k}$, where

$$\chi_{1k} = \{1 - 2(1-\xi)d_k\}v_k^\top\Xi v_k, \qquad \chi_{2k} = \{1 - 2(1-\xi)c_k\}u_k^\top\Xi u_k, \tag{8}$$

$$\psi_{1k} = -\{1 - 2(1-\xi)d_k\}M_{1k}\Xi v_k - (1-\xi)(v_k^\top\Xi v_k)v_k, \tag{9}$$

$$\psi_{2k} = -\{1 - 2(1-\xi)c_k\}M_{2k}\Xi u_k - (1-\xi)(u_k^\top\Xi u_k)u_k, \tag{10}$$

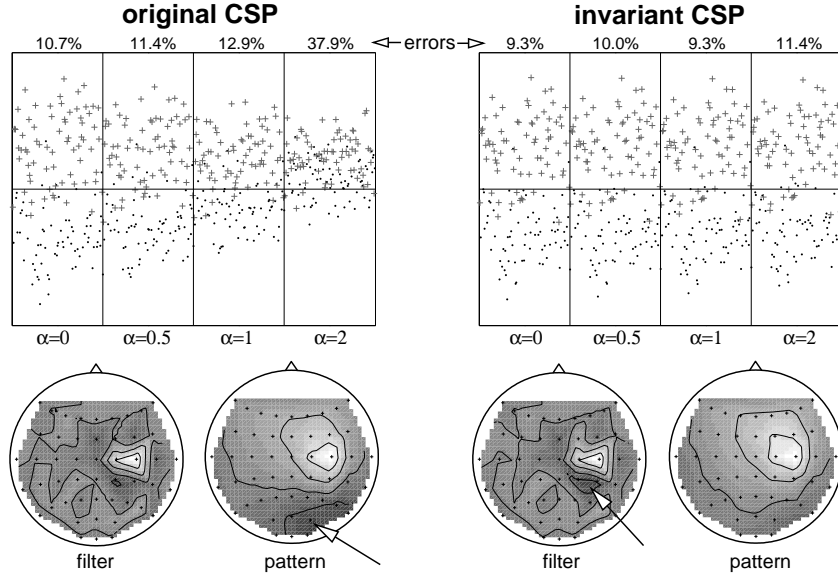

Figure 2: Comparison of CSP and iCSP on test data with artificially increased occipital alpha. The upper plots show the classifier output on the test data with different degrees of alpha added (factors $\alpha$= 0, 0.5, 1, 2). The lower panel shows the filter/pattern coefficients topographically mapped on the scalp from original CSP (left) and iCSP (right). Here the invariance property was defined with respect to the increase in the alpha activity in the visual cortex (occipital location) using an eyes open/eyes closed recording. See Section 3 for the definition of filter and pattern.

$M_{1k} := \Sigma^{-1/2}(\Sigma^{-1/2}\Sigma_1\Sigma^{-1/2} - d_kI)^+\Sigma^{-1/2}$, $M_{2k} := \Sigma^{-1/2}(\Sigma^{-1/2}\Sigma_2\Sigma^{-1/2} - d_kI)^+\Sigma^{-1/2}$, and $\Sigma := (1-\xi)(\Sigma_1 + \Sigma_2) + \xi\Xi$. The implication of the result is the following. If $\xi = 1 - \frac{1}{2d_k}$ (resp. $\xi = 1 - \frac{1}{2c_k}$) is satisfied, the $O(\varepsilon)$ term $\chi_{1k}$ (resp. $\chi_{2k}$) of the $k$-th eigenvalue vanishes and also the $k$-th eigenvector does coincide with the one for the original problem up to $\varepsilon$ order, because the first term of $\psi_{1k}$ (resp. $\psi_{2k}$) becomes zero (we note that $d_k$ and $c_k$ also depend on $\xi$).

## 4   Evaluation

**Test Case with Constructed Test Data.**   To validate the proposed iCSP, we first applied it to specifically constructed test data. iCSP was trained ($\xi = 0.5$) on motor imagery data with the invariance characterized by data from a measurement during 'eyes open' (approx. 40 s) and 'eyes closed' (approx. 20 s). The motor imagery test data was used in its original form and variants that were modified in a controlled manner: From another data set during 'eyes closed' we extracted activity related to increased occipital alpha activity (backprojection of 5 ICA components) and added this with 3 different factors ($\alpha = 0.5$, 1, 2) to the test data.

The upper plots of Fig. 2 display the classifier output on the constructed test data. While the performance of the original CSP is more and more deteriorated with increased alpha mixed in, the proposed iCSP method maintains a stable performance independent of the amount of increased alpha activity. The spatial filters that were extracted by CSP analysis vs. the proposed iCSP often look quite similar. However, tiny but apparently important differences exist. In the lower panel of Fig. 2 the filter ($v_j$) pattern ($a_j$) pairs from original CSP (left) and iCSP (right) are shown. The *filters* from two approaches resemble each other strongly. However, the corresponding *patterns* reveal an important difference. While the pattern of the original CSP has positive weights at the right occipital side which might be susceptible to $\alpha$ modulations, the corresponding iCSP has not. A more detailed inspection shows that both filters have a focus over the right (sensori-) motor cortex, but only the invariant filter has a spot of opposite sign right posterior to it. This spot will filter out contributions coming from occipital/parietal sites.

**Model selection for iCSP.**   For each subject, a cross-validation was performed for different values of $\xi$ on the training data (session *imag_move*) and the $\xi$ resulting in minimum error was chosen. For the same values of $\xi$ the iCSP filters + LDA classifier trained on *imag_move* were applied to calcu-

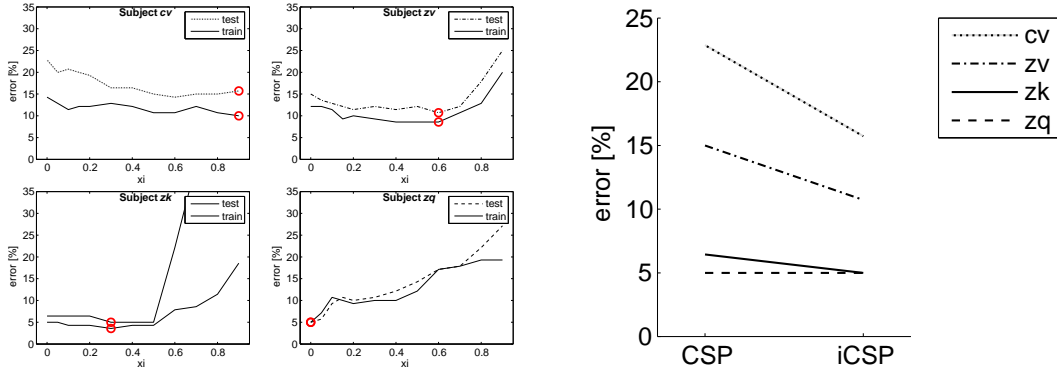

Figure 3: Modelselection and evaluation. *Left subplots:* Selection of hyperparameter $\xi$ of the iCSP method. For each subject, a cross-validation was performed for different values of $\xi$ on the training data (session *imag_move*), see thin black line, and the $\xi$ resulting in minimum error was chosen (circle). For the same values of $\xi$ the iCSP filters + LDA classifier trained on *imag_move* were applied to calculate the test error on data from *imag_lett* (thick colorful line). *Right plot:* Test error in all four recordings for classical CSP and the proposed iCSP (with model parameter $\xi$ chosen by cross-validation on the training set as described in Section 4).

late the test error on data from *imag_lett*. Fig. 3 (left plots) shows the result of this procedure. The shape of the cross-validation error on the training set and the test error is very similar. Accordingly, the selection of values for parameter $\xi$ is successful. For subject *zq* $\xi = 0$ was chosen, i.e. classical CSP. The case for subject *zk* shows that the selection of $\xi$ may be a delicate issue. For larges values of $\xi$ cross-validation error and test error differ dramatically. A choice of $\xi > 0.5$ would result in bad performance of iCSP, while this effect could have not been predicted so severely from the cross-validation of the training set.

**Evaluation of Performance with Real BCI Data.** For evaluation we used the *imag_move* session (see Section 2) as training set and the *imag_lett* session as test set. Fig 3 (right plot) compares the classification error obtained by classical CSP and by the proposed method iCSP with model parameter $\xi$ chosen by cross-validation on the training set as described above. Again an excellent improvement is visible.

## 5   Concluding discussion

EEG data from Brain-Computer Interface experiments are highly challenging to evaluate due to noise, nonstationarity and diverse artifacts. Thus, BCI provides an excellent testbed for testing the quality and applicability of robust machine learning methods (cf. the BCI Competitions [27, 28]).

Obviously BCI users are subject to variations in attention and motivation. These types of non-stationarities can considerably deteriorate the BCI classifier performance. In present paper we proposed a novel method to alleviate this problem.

A limitation of our method is that variations need to be characterized in advance (by estimating an appropriate covariance matrix). At the same time this is also a strength of our method as neurophysiological prior knowledge about possible sources of non-stationarity is available and can thus be taken into account in a controlled manner. Also the selection of hyperparameter $\xi$ needs more investigation, cf. the case of subject *zk* in Fig. 3. One strategy to pursue is to update the covariance matrix $\Xi$ online with incoming test data. (Note that no label information is needed.) Online learning (learning algorithms for adaptation within a BCI session) could also be used to further stabilize the system against unforeseen changes. It remains to future research to explore this interesting direction.

## Appendix: Proof of Lemma 1.

By substituting the expansions of $\widetilde{\lambda}_k$ and $\widetilde{w}_k$ to Eq.(6) and taking the $O(\varepsilon)$ term, we get

$$A\psi_k + \Delta w_k = \lambda_k B\psi_k + \lambda_k P w_k + \chi_k B w_k. \tag{11}$$

Eq.(7) can be obtained by multiplying $w_k^\top$ to Eq.(11) and applying Eq.(5). Then, from Eq.(11),

$$(A - \lambda_k B)\psi_k \; = \; -(\Delta - \lambda_k P)w_k + \chi_k B w_k \; = \; -(A - \lambda_k B)M_k(\Delta - \lambda_k P)w_k,$$

holds, where we used the constraints $w_j^\top Bw_k = \delta_{jk}$ and

$$(A - \lambda_k B)M_k = \sum_{j \neq k} Bw_j w_j^\top = I - Bw_k w_k^\top. \tag{12}$$

Eq.(12) can be proven by $B^{-1/2}AB^{-1/2} - \lambda_k I = \sum_{j \neq k} \lambda_j B^{1/2} w_j w_j^\top B^{1/2}$ and $(B^{-1/2}AB^{-1/2} - \lambda_k I)^+ = \sum_{j \neq k} 1/\lambda_j B^{1/2} w_j w_j^\top B^{1/2}$. Since span$\{w_k\}$ is the kernel of the operator $A - \lambda_k B$, $\psi_k$ can be explained as $\psi_k = -M_k(\Delta - \lambda_k P)w_k + cw_k$. By a multiplication with $w_k^\top B$, the constant $c$ turns out to be $c = -w_k^\top Pw_k/2$, where we used the fact $w_k^\top BM_k = \mathbf{0}^\top$ and $w_k^\top B\psi_k = -w_k^\top Pw_k/2$ derived from the normalization $\widetilde{w}_k^\top(B + \varepsilon P)\widetilde{w}_k = 1$. $\qquad\square$

## Footnotes

[1]Note: In our exposition we focus on EEG-based BCI systems that does not rely on evoked potentials (for an extensive overview of BCI systems including invasive and systems based on evoked potentials see [1]).

[2]We use the term covariance for zero-delay second order statistics between channels and not for the statistical variability. Since we assume the signal to be band-pass filtered, the second order statistics reflects band power.

## References

[1] J. R. Wolpaw, N. Birbaumer, D. J. McFarland, G. Pfurtscheller, and T. M. Vaughan, "Brain-computer interfaces for communication and control", *Clin. Neurophysiol.*, 113: 767–791, 2002.

[2] N. Birbaumer, N. Ghanayim, T. Hinterberger, I. Iversen, B. Kotchoubey, A. Kübler, J. Perelmouter, E. Taub, and H. Flor, "A spelling device for the paralysed", *Nature*, 398: 297–298, 1999.

[3] G. Pfurtscheller, C. Neuper, C. Guger, W. Harkam, R. Ramoser, A. Schlögl, B. Obermaier, and M. Pregenzer, "Current Trends in Graz Brain-computer Interface (BCI)", *IEEE Trans. Rehab. Eng.*, 8(2): 216–219, 2000.

[4] J. Millán, *Handbook of Brain Theory and Neural Networks*, MIT Press, Cambridge, 2002.

[5] E. A. Curran and M. J. Stokes, "Learning to control brain activity: A review of the production and control of EEG components for driving brain-computer interface (BCI) systems", *Brain Cogn.*, 51: 326–336, 2003.

[6] G. Dornhege, J. del R. Millán, T. Hinterberger, D. McFarland, and K.-R. Müller, eds., *Toward Brain-Computer Interfacing*, MIT Press, Cambridge, MA, 2007.

[7] T. Elbert, B. Rockstroh, W. Lutzenberger, and N. Birbaumer, "Biofeedback of Slow Cortical Potentials. I", *Electroencephalogr. Clin. Neurophysiol.*, 48: 293–301, 1980.

[8] C. Guger, H. Ramoser, and G. Pfurtscheller, "Real-time EEG analysis with subject-specific spatial patterns for a Brain Computer Interface (BCI)", *IEEE Trans. Neural Sys. Rehab. Eng.*, 8(4): 447–456, 2000.

[9] B. Blankertz, G. Curio, and K.-R. Müller, "Classifying Single Trial EEG: Towards Brain Computer Interfacing", in: T. G. Diettrich, S. Becker, and Z. Ghahramani, eds., *Advances in Neural Inf. Proc. Systems (NIPS 01)*, vol. 14, 157–164, 2002.

[10] L. Parra, C. Alvino, A. C. Tang, B. A. Pearlmutter, N. Yeung, A. Osman, and P. Sajda, "Linear spatial integration for single trial detection in encephalography", *NeuroImage*, 7(1): 223–230, 2002.

[11] E. Curran, P. Sykacek, S. Roberts, W. Penny, M. Stokes, I. Jonsrude, and A. Owen, "Cognitive tasks for driving a Brain Computer Interfacing System: a pilot study", *IEEE Trans. Rehab. Eng.*, 12(1): 48–54, 2004.

[12] J. Millán, F. Renkens, J. M. no, and W. Gerstner, "Non-invasive brain-actuated control of a mobile robot by human EEG", *IEEE Trans. Biomed. Eng.*, 51(6): 1026–1033, 2004.

[13] N. J. Hill, T. N. Lal, M. Schröder, T. Hinterberger, B. Wilhelm, F. Nijboer, U. Mochty, G. Widman, C. E. Elger, B. Schölkopf, A. Kübler, and N. Birbaumer, "Classifying EEG and ECoG Signals without Subject Training for Fast BCI Implementation: Comparison of Non-Paralysed and Completely Paralysed Subjects", *IEEE Trans. Neural Sys. Rehab. Eng.*, 14(6): 183–186, 2006.

[14] B. Blankertz, G. Dornhege, M. Krauledat, K.-R. Müller, and G. Curio, "The non-invasive Berlin Brain-Computer Interface: Fast Acquisition of Effective Performance in Untrained Subjects", *NeuroImage*, 37(2): 539–550, 2007, URL http://dx.doi.org/10.1016/j.neuroimage.2007.01.051.

[15] B. Blankertz, R. Tomioka, S. Lemm, M. Kawanabe, and K.-R. Müller, "Optimizing Spatial Filters for Robust EEG Single-Trial Analysis", *IEEE Signal Proc. Magazine*, 25(1): 41–56, 2008, URL http://dx.doi.org/10.1109/MSP.2008.4408441.

[16] S. Mika, G. Rätsch, J. Weston, B. Schölkopf, A. Smola, and K.-R. Müller, "Invariant Feature Extraction and Classification in Kernel Spaces", in: S. Solla, T. Leen, and K.-R. Müller, eds., *Advances in Neural Information Processing Systems*, vol. 12, 526–532, MIT Press, 2000.

[17] H. Berger, "Über das Elektroenkephalogramm des Menschen", *Archiv für Psychiatrie und Nervenkrankheiten*, 99(6): 555–574, 1933.

[18] H. Jasper and H. Andrews, "Normal differentiation of occipital and precentral regions in man", *Arch. Neurol. Psychiat. (Chicago)*, 39: 96–115, 1938.

[19] G. Pfurtscheller and F. H. L. da Silva, "Event-related EEG/MEG synchronization and desynchronization: basic principles", *Clin. Neurophysiol.*, 110(11): 1842–1857, 1999.

[20] Z. J. Koles, "The quantitative extraction and topographic mapping of the abnormal components in the clinical EEG", *Electroencephalogr. Clin. Neurophysiol.*, 79(6): 440–447, 1991.

[21] K. Fukunaga, *Introduction to statistical pattern recognition*, Academic Press, Boston, 2nd edn., 1990.

[22] B. Schölkopf, *Support vector learning*, Oldenbourg Verlag, Munich, 1997.

[23] G. Dornhege, B. Blankertz, G. Curio, and K.-R. Müller, "Boosting bit rates in non-invasive EEG single-trial classifications by feature combination and multi-class paradigms", *IEEE Trans. Biomed. Eng.*, 51(6): 993–1002, 2004.

[24] F. R. Hampel, E. M. Ronchetti, P. J. Rousseeuw, and W. A. Stahel, *Robust Statistics: The Approach Based on Influence Functions*, Wiley, New York, 1986.

[25] F. Critchley, "Influence in principal components analysis", *Biometrika*, 72(3): 627–636, 1985.

[26] M. Romanazzi, "Influence in Canonical Correlation Analysis", *Psychometrika*, 57(2): 237–259, 1992.

[27] B. Blankertz, K.-R. Müller, G. Curio, T. M. Vaughan, G. Schalk, J. R. Wolpaw, A. Schlögl, C. Neuper, G. Pfurtscheller, T. Hinterberger, M. Schröder, and N. Birbaumer, "The BCI Competition 2003: Progress and Perspectives in Detection and Discrimination of EEG Single Trials", *IEEE Trans. Biomed. Eng.*, 51(6): 1044–1051, 2004.

[28] B. Blankertz, K.-R. Müller, D. Krusienski, G. Schalk, J. R. Wolpaw, A. Schlögl, G. Pfurtscheller, J. del R. Millán, M. Schröder, and N. Birbaumer, "The BCI Competition III: Validating Alternative Approachs to Actual BCI Problems", *IEEE Trans. Neural Sys. Rehab. Eng.*, 14(2): 153–159, 2006.

